# Statistical Dynamics of Batch Learning

**S. Li and K. Y. Michael Wong**
Department of Physics, Hong Kong University of Science and Technology
Clear Water Bay, Kowloon, Hong Kong
{*phlisong, phkywong*} *@ust.hk*

## Abstract

An important issue in neural computing concerns the description of learning dynamics with macroscopic dynamical variables. Recent progress on *on-line* learning only addresses the often unrealistic case of an infinite training set. We introduce a new framework to model batch learning of restricted sets of examples, widely applicable to *any* learning cost function, and fully taking into account the temporal correlations introduced by the recycling of the examples. For illustration we analyze the effects of weight decay and early stopping during the learning of teacher-generated examples.

## 1  Introduction

The dynamics of learning in neural computing is a complex multi-variate process. The interest on the macroscopic level is thus to describe the process with macroscopic dynamical variables. Recently, much progress has been made on modeling the dynamics of *on-line* learning, in which an independent example is generated for each learning step [1, 2]. Since statistical correlations among the examples can be ignored, the dynamics can be simply described by instantaneous dynamical variables.

However, most studies on on-line learning focus on the ideal case in which the network has access to an almost infinite training set, whereas in many applications, the collection of training examples may be costly. A restricted set of examples introduces extra temporal correlations during learning, and the dynamics is much more complicated. Early studies briefly considered the dynamics of Adaline learning [3, 4, 5], and has recently been extended to *linear* perceptrons learning nonlinear rules [6, 7]. Recent attempts, using the *dynamical replica theory*, have been made to study the learning of restricted sets of examples, but so far exact results are published for simple learning rules such as Hebbian learning, beyond which appropriate approximations are needed [8].

In this paper, we introduce a new framework to model batch learning of restricted sets of examples, widely applicable to any learning rule which minimizes an *arbitrary* cost function by gradient descent. It fully takes into account the temporal correlations during learning, and is therefore exact for large networks.

## 2 Formulation

Consider the single layer perceptron with $N \gg 1$ input nodes $\{\xi_j\}$ connecting to a single output node by the weights $\{J_j\}$. For convenience we assume that the inputs $\xi_j$ are Gaussian variables with mean 0 and variance 1, and the output state $S$ is a function $f(x)$ of the *activation* $x$ at the output node, i.e.

$$S = f(x); \quad x = \vec{J} \cdot \vec{\xi}. \tag{1}$$

The network is assigned to "learn" $p \equiv \alpha N$ examples which map inputs $\{\xi_j^\mu\}$ to the outputs $\{S_\mu\}$ $(\mu = 1, \ldots, p)$. $S_\mu$ are the outputs generated by a teacher perceptron $\{B_j\}$, namely

$$S_\mu = f(y_\mu); \quad y_\mu = \vec{B} \cdot \vec{\xi}^\mu. \tag{2}$$

Batch learning by gradient descent is achieved by adjusting the weights $\{J_j\}$ iteratively so that a certain cost function in terms of the student and teacher activations $\{x_\mu\}$ and $\{y_\mu\}$ is minimized. Hence we consider a general cost function

$$E = -\sum_\mu g(x_\mu, y_\mu). \tag{3}$$

The precise functional form of $g(x, y)$ depends on the adopted learning algorithm. For the case of binary outputs, $f(x) = \mathrm{sgn}\, x$. Early studies on the learning dynamics considered Adaline learning [3, 4, 5], where $g(x, y) = -(S - x)^2/2$ with $S = \mathrm{sgn}\, y$. For recent studies on Hebbian learning [8], $g(x, y) = xS$.

To ensure that the perceptron is regularized after learning, it is customary to introduce a weight decay term. Furthermore, to avoid the system being trapped in local minima, noise is often added in the dynamics. Hence the gradient descent dynamics is given by

$$\frac{dJ_j(t)}{dt} = \frac{1}{N} \sum_\mu g'(x_\mu(t), y_\mu)\xi_j^\mu - \lambda J_j(t) + \eta_j(t), \tag{4}$$

where, here and below, $g'(x, y)$ and $g''(x, y)$ respectively represent the first and second partial derivatives of $g(x, y)$ with respect to $x$. $\lambda$ is the weight decay strength, and $\eta_j(t)$ is the noise term at temperature $T$ with

$$\langle \eta_j(t) \rangle = 0 \quad \text{and} \quad \langle \eta_j(t)\eta_k(s) \rangle = \frac{2T}{N}\delta_{jk}\delta(t - s). \tag{5}$$

## 3 The Cavity Method

Our theory is the dynamical version of the cavity method [9, 10, 11]. It uses a self-consistency argument to consider what happens when a new example is added to a training set. The central quantity in this method is the *cavity activation*, which is the activation of a new example for a perceptron trained without that example. Since the original network has no information about the new example, the cavity activation is stochastic. Specifically, denoting the new example by the label 0, its cavity activation at time $t$ is

$$h_0(t) = \vec{J}(t) \cdot \vec{\xi}^0. \tag{6}$$

For large $N$ and independently generated examples, $h_0(t)$ is a Gaussian variable. Its covariance is given by the correlation function $C(t, s)$ of the weights at times $t$ and $s$, that is,

$$\langle h_0(t)h_0(s) \rangle = \vec{J}(t) \cdot \vec{J}(s) \equiv C(t, s), \tag{7}$$

where $\xi_j^0$ and $\xi_k^0$ are assumed to be independent for $j \neq k$. The distribution is further specified by the teacher-student correlation $R(t)$, given by

$$\langle h_0(t)y_0 \rangle = \vec{J}(t) \cdot \vec{B} \equiv R(t). \tag{8}$$

Now suppose the perceptron incorporates the new example at the batch-mode learning step at time $s$. Then the activation of this new example at a subsequent time $t > s$ will no longer be a random variable. Furthermore, the activations of the original $p$ examples at time $t$ will also be adjusted from $\{x_\mu(t)\}$ to $\{x_\mu^0(t)\}$ because of the newcomer, which will in turn affect the evolution of the activation of example 0, giving rise to the so-called Onsager reaction effects. This makes the dynamics complex, but fortunately for large $p \sim N$, we can assume that the adjustment from $x_\mu(t)$ to $x_\mu^0(t)$ is small, and perturbative analysis can be applied.

Suppose the weights of the original and new perceptron at time $t$ are $\{J_j(t)\}$ and $\{J_j^0(t)\}$ respectively. Then a perturbation of (4) yields

$$\begin{aligned}
\left(\frac{d}{dt} + \lambda\right)(J_j^0(t) - J_j(t)) &= \frac{1}{N}g'(x_0(t), y_0)\xi_j^0 \\
&+ \frac{1}{N}\sum_{\mu k}\xi_j^\mu g''(x_\mu(t), y_\mu)\xi_k^\mu(J_k^0(t) - J_k(t)). \tag{9}
\end{aligned}$$

The first term on the right hand side describes the primary effects of adding example 0 to the training set, and is the driving term for the difference between the two perceptrons. The second term describes the secondary effects due to the changes to the original examples caused by the added example, and is referred to as the Onsager reaction term. One should note the difference between the cavity and generic activations of the added example. The former is denoted by $h_0(t)$ and corresponds to the activation in the perceptron $\{J_j(t)\}$, whereas the latter, denoted by $x_0(t)$ and corresponding to the activation in the perceptron $\{J_j^0(t)\}$, is the one used in calculating the gradient in the driving term of (9). Since their notations are sufficiently distinct, we have omitted the superscript 0 in $x_0(t)$, which appears in the background examples $x_\mu^0(t)$.

The equation can be solved by the Green's function technique, yielding

$$J_j^0(t) - J_j(t) = \sum_k \int ds\, G_{jk}(t, s)\left(\frac{1}{N}g_0'(s)\xi_k^0\right), \tag{10}$$

where $g_0'(s) = g'(x_0(s), y_0)$ and $G_{jk}(t, s)$ is the *weight Green's function* satisfying

$$G_{jk}(t, s) = G^{(0)}(t - s)\delta_{jk} + \frac{1}{N}\sum_{\mu i}\int dt'\, G^{(0)}(t - t')\xi_j^\mu g_\mu''(t')\xi_i^\mu G_{ik}(t' - s), \tag{11}$$

$G^{(0)}(t - s) \equiv \Theta(t - s)\exp(-\lambda(t - s))$ is the bare Green's function, and $\Theta$ is the step function. The weight Green's function describes how the effects of example 0 propagates from weight $J_k$ at learning time $s$ to weight $J_j$ at a subsequent time $t$, including both primary and secondary effects. Hence all the temporal correlations have been taken into account.

For large $N$, the equation can be solved by a diagrammatic approach similar to [5]. The weight Green's function is self-averaging over the distribution of examples and is diagonal, i.e. $\lim_{N\to\infty} G_{jk}(t, s) = G(t, s)\delta_{jk}$, where

$$G(t, s) = G^{(0)}(t - s) + \alpha \int dt_1 \int dt_2\, G^{(0)}(t - t_1)\langle g_\mu''(t_1)D_\mu(t_1, t_2)\rangle G(t_2, s). \tag{12}$$

$D_\mu(t,s)$ is the *example Green's function* given by

$$D_\mu(t,s) = \delta(t-s) + \int dt' G(t,t') g_\mu''(t') D_\mu(t',s).$$  (13)

This allows us to express the generic activations of the examples in terms of their cavity counterparts. Multiplying both sides of (10) and summing over $j$, we get

$$x_0(t) - h_0(t) = \int ds G(t,s) g_0'(s).$$  (14)

This equation is interpreted as follows. At time $t$, the generic activation $x_0(t)$ deviates from its cavity counterpart because its gradient term $g_0'(s)$ was present in the batch learning step at previous times $s$. This gradient term propagates its influence from time $s$ to $t$ via the Green's function $G(t,s)$. Statistically, this equation enables us to express the activation distribution in terms of the cavity activation distribution, thereby getting a macroscopic description of the dynamics.

To solve for the Green's functions and the activation distributions, we further need the fluctuation-response relation derived by linear response theory,

$$C(t,s) = \alpha \int dt' G^{(0)}(t-t')\langle g_\mu'(t') x_\mu(s)\rangle + 2T \int dt' G^{(0)}(t-t') G(s,t').$$  (15)

Finally, the teacher-student correlation is given by

$$R(t) = \alpha \int dt' G^{(0)}(t-t')\langle g_\mu'(t') y_\mu\rangle.$$  (16)

## 4 A Solvable Case

The cavity method can be applied to the dynamics of learning with an arbitrary cost function. When it is applied to the Hebb rule, it yields results identical to the exact results in [8]. Here we present the results for the Adaline rule to illustrate features of learning dynamics derivable from the study. This is a common learning rule and bears resemblance with the more common back-propagation rule. Theoretically, its dynamics is particularly convenient for analysis since $g''(x) = -1$, rendering the weight Green's function time translation invariant, i.e. $G(t,s) = G(t-s)$. In this case, the dynamics can be solved by Laplace transform.

To monitor the progress of learning, we are interested in three performance measures: (a) *Training error* $\epsilon_t$, which is the probability of error for the training examples. It is given by $\epsilon_t = \langle \Theta(-x \text{sgn} y)\rangle_{xy}$, where the average is taken over the joint distribution $p(x,y)$ of the training set. (b) *Test error* $\epsilon_{test}$, which is the probability of error when the inputs $\xi_j^\mu$ of the training examples are corrupted by an additive Gaussian noise of variance $\Delta^2$. This is a relevant performance measure when the perceptron is applied to process data which are the corrupted versions of the training data. It is given by $\epsilon_{test} = \langle H(x \text{sgn} y/\Delta \sqrt{C(t,t)})\rangle_{xy}$. When $\Delta^2 = 0$, the test error reduces to the training error. (c) *Generalization error* $\epsilon_g$, which is the probability of error for an arbitrary input $\xi_j$ when the teacher and student outputs are compared. It is given by $\epsilon_g = \arccos[R(t)/\sqrt{C(t,t)}]/\pi$.

Figure 1(a) shows the evolution of the generalization error at $T = 0$. When the weight decay strength varies, the steady-state generalization error is minimized at the optimum

$$\lambda_{opt} = \frac{\pi}{2} - 1,$$  (17)

which is independent of $\alpha$. It is interesting to note that in the cases of the linear perceptron, the optimal weight decay strength is also independent of $\alpha$ and only determined by the output noise and unlearnability of the examples [5, 7]. Similarly, here the student is only provided the coarse-grained version of the teacher's activation in the form of binary bits.

For $\lambda < \lambda_{opt}$, the generalization error is a non-monotonic function in learning time. Hence the dynamics is plagued by *overtraining*, and it is desirable to introduce *early stopping* to improve the perceptron performance. Similar behavior is observed in linear perceptrons [5, 6, 7].

To verify the theoretical predictions, simulations were done with $N = 500$ and using 50 samples for averaging. As shown in Fig. 1(a), the agreement is excellent.

Figure 1(b) compares the generalization errors at the steady-state and the early stopping point. It shows that early stopping improves the performance for $\lambda < \lambda_{opt}$, which becomes near-optimal when compared with the best result at $\lambda = \lambda_{opt}$. Hence early stopping can speed up the learning process without significant sacrifice in the generalization ability. However, it cannot outperform the optimal result at steady-state. This agrees with a recent empirical observation that a careful control of the weight decay may be better than early stopping in optimizing generalization [12].

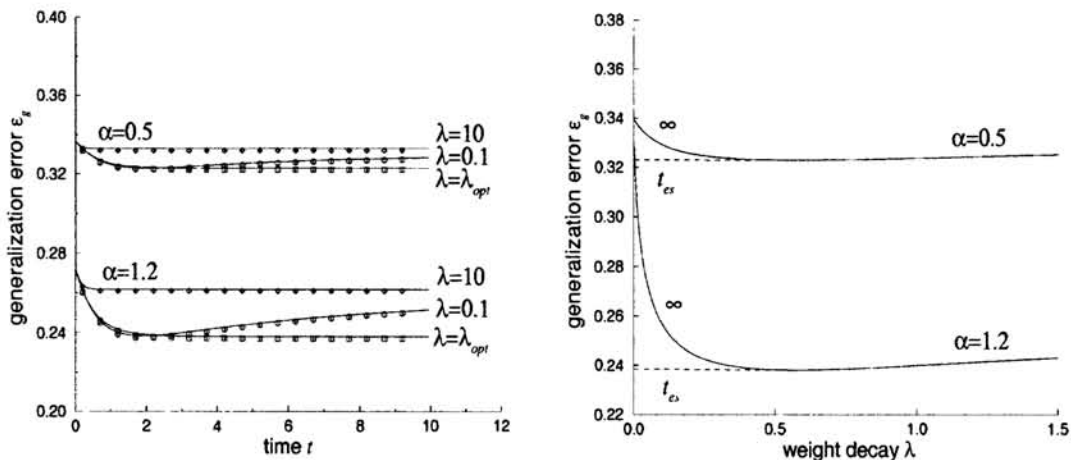

Figure 1: (a) The evolution of the generalization error at $T = 0$ for $\alpha = 0.5, 1.2$ and different weight decay strengths $\lambda$. Theory: solid line, simulation: symbols. (b) Comparing the generalization error at the steady state ($\infty$) and at the early stopping point ($t_{es}$) for $\alpha = 0.5, 1.2$ and $T = 0$.

In the search for optimal learning algorithms, an important consideration is the environment in which the performance is tested. Besides the generalization performance, there are applications in which the test examples have inputs correlated with the training examples. Hence we are interested in the evolution of the test error for a given additive Gaussian noise $\Delta$ in the inputs. Figure 2(a) shows, again, that there is an optimal weight decay parameter $\lambda_{opt}$ which minimizes the test error. Furthermore, when the weight decay is weak, early stopping is desirable.

Figure 2(b) shows the value of the optimal weight decay as a function of the input noise variance $\Delta^2$. To the lowest order approximation, $\lambda_{opt} \propto \Delta^2$ for sufficiently large $\Delta^2$. The dependence of $\lambda_{opt}$ on input noise is rather general since it also holds in the case of random examples [13]. In the limit of small $\Delta^2$, $\lambda_{opt}$ vanishes as $\Delta^2$ for $\alpha < 1$, whereas $\lambda_{opt}$ approaches a nonzero constant for $\alpha > 1$. Hence for

$\alpha < 1$, weight decay is not necessary when the training error is optimized, but when the perceptron is applied to process increasingly noisy data, weight decay becomes more and more important in performance enhancement.

Figure 2(b) also shows the phase line $\lambda_{ot}(\Delta^2)$ below which overtraining occurs. Again, to the lowest order approximation, $\lambda_{ot} \propto \Delta^2$ for sufficiently large $\Delta^2$. However, unlike the case of generalization error, the line for the onset of overtraining does not coincide exactly with the line of optimal weight decay. In particular, for an intermediate range of input noise, the optimal line lies in the region of overtraining, so that the optimal performance can only be attained by tuning *both* the weight decay strength and learning time. However, at least in the present case, computational results show that the improvement is marginal.

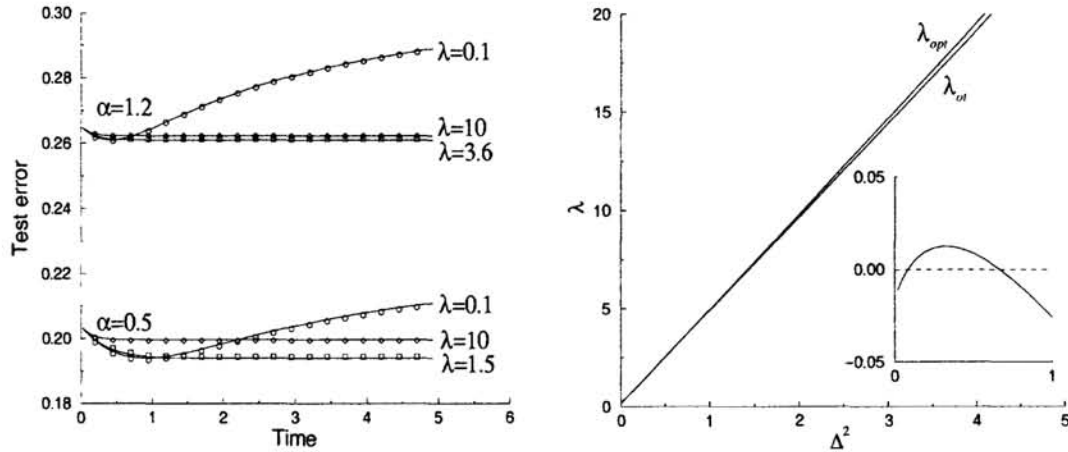

Figure 2: (a) The evolution of the test error for $\Delta^2 = 3$, $T = 0$ and different weight decay strengths $\lambda$ ($\lambda_{opt} \approx 1.5, 3.6$ for $\alpha = 0.5, 1.2$ respectively). (b) The lines of the optimal weight decay and the onset of overtraining for $\alpha = 5$. Inset: The same data with $\lambda_{ot} - \lambda_{opt}$ (magnified) versus $\Delta^2$.

## 5 Conclusion

Based on the cavity method, we have introduced a new framework for modeling the dynamics of learning, which is applicable to *any* learning cost function, making it a versatile theory. It takes into full account the temporal correlations generated by the use of a restricted set of examples, which is more realistic in many situations than theories of on-line learning with an infinite training set.

While the Adaline rule is solvable by the cavity method, it is still a relatively simple model approachable by more direct methods. Hence the justification of the method as a general framework for learning dynamics hinges on its applicability to less trivial cases. In general, $g''_\mu(t')$ in (13) is not a constant and $D_\mu(t, s)$ has to be expanded as a series. The dynamical equations can then be considered as the starting point of a perturbation theory, and results in various limits can be derived, e.g. the limits of small $\alpha$, large $\alpha$, large $\lambda$, or the asymptotic limit. Another area for the useful application of the cavity method is the case of batch learning with very large learning steps. Since it has been shown recently that such learning converges in a few steps [6], the dynamical equations remain simple enough for a meaningful study. Preliminary results along this direction are promising and will be reported elsewhere.

An alternative general theory for learning dynamics, the dynamical replica theory, has recently been developed [8]. It yields exact results for Hebbian learning, and approximate results for more non-trivial cases. Based on certain self-averaging assumptions, the theory is able to approximate the dynamics by the evolution of single-time functions, at the expense of having to solve a set of saddle point equations in the replica formalism at every learning instant. On the other hand, our theory retains the functions $G(t, s)$ and $C(t, s)$ with double arguments, but develops naturally from the stochastic nature of the cavity activations. Contrary to a suggestion [14], the cavity method can also be applied to the on-line learning with restricted sets of examples. It is hoped that by adhering to an exact formalism, the cavity method can provide more fundamental insights when the studies are extended to more sophisticated multilayer networks of practical importance.

The method enables us to study the effects of weight decay and early stopping. It shows that the optimal strength of weight decay is determined by the imprecision in the examples, or the level of input noise in anticipated applications. For weaker weight decay, the generalization performance can be made near-optimal by early stopping. Furthermore, depending on the performance measure, optimality may only be attained by a combination of weight decay and early stopping. Though the performance improvement is marginal in the present case, the question remains open in the more general context.

We consider the present work as the beginning of an in-depth study of learning dynamics. Many interesting and challenging issues remain to be explored.

## Acknowledgments

We thank A. C. C. Coolen and D. Saad for fruitful discussions during NIPS. This work was supported by the grant HKUST6130/97P from the Research Grant Council of Hong Kong.

## References

[1] D. Saad and S. Solla, *Phys. Rev. Lett.* **74**, 4337 (1995).

[2] D. Saad and M. Rattray, *Phys. Rev. Lett.* **79**, 2578 (1997).

[3] J. Hertz, A. Krogh and G. I. Thorbergssen, *J. Phys. A* **22**, 2133 (1989).

[4] M. Opper, *Europhys. Lett.* **8**, 389 (1989).

[5] A. Krogh and J. A. Hertz, *J. Phys. A* **25**, 1135 (1992).

[6] S. Bös and M. Opper, *J. Phys. A* **31**, 4835 (1998).

[7] S. Bös, *Phys. Rev. E* **58**, 833 (1998).

[8] A. C. C. Coolen and D. Saad, in *On-line Learning in Neural Networks*, ed. D. Saad (Cambridge University Press, Cambridge, 1998).

[9] M. Mézard, G. Parisi and M. Virasoro, *Spin Glass Theory and Beyond* (World Scientific, Singapore) (1987).

[10] K. Y. M. Wong, *Europhys. Lett.* **30**, 245 (1995).

[11] K. Y. M. Wong, *Advances in Neural Information Processing Systems* **9**, 302 (1997).

[12] L. K. Hansen, J. Larsen and T. Fog, *IEEE Int. Conf. on Acoustics, Speech, and Signal Processing* **4**, 3205 (1997).

[13] Y. W. Tong, K. Y. M. Wong and S. Li, to appear in *Proc. of IJCNN'99* (1999).

[14] A. C. C. Coolen and D. Saad, Preprint KCL-MTH-99-33 (1999).